# Projection Retrieval for Classification

**Madalina Fiterau**
Machine Learning Department
Carnegie Mellon University
Pittsburgh, PA 15213
mfiterau@cs.cmu.edu

**Artur Dubrawski**
School of Computer Science
Carnegie Mellon University
Pittsburgh, PA 15213
awd@cs.cmu.edu

## Abstract

In many applications, classification systems often require human intervention in the loop. In such cases the decision process must be transparent and comprehensible, simultaneously requiring minimal assumptions on the underlying data distributions. To tackle this problem, we formulate an axis-aligned subspace-finding task under the assumption that query specific information dictates the complementary use of the subspaces. We develop a regression-based approach called RECIP that efficiently solves this problem by finding projections that minimize a nonparametric conditional entropy estimator. Experiments show that the method is accurate in identifying the informative projections of the dataset, picking the correct views to classify query points, and facilitates visual evaluation by users.

## 1 Introduction and problem statement

In the domain of predictive analytics, many applications which keep human users in the loop require the use of simple classification models. Often, it is required that a test-point be 'explained' (classified) using a simple low-dimensional projection of the original feature space. This is a Projection Retrieval for Classification problem (PRC). The interaction with the user proceeds as follows: the user provides the system a query point; the system searches for a projection in which the point can be accurately classified; the system displays the classification result as well as an illustration of how the classification decision was reached in the selected projection.

Solving the PRC problem is relevant in many practical applications. For instance, consider a nuclear threat detection system installed at a border check point. Vehicles crossing the border are scanned with sensors so that a large array of measurements of radioactivity and secondary contextual information is being collected. These observations are fed into a classification system that determines whether the scanned vehicle may carry a threat. Given the potentially devastating consequences of a false negative, a border control agent is requested to validate the prediction and decide whether to submit the vehicle for a costly further inspection. With the positive classification rate of the system under strict bounds because of limitations in the control process, the risk of false negatives is increased. Despite its crucial role, human intervention should only be withheld for cases in which there are reasons to doubt the validity of classification. In order for a user to attest the validity of a decision, the user must have a good understanding of the classification process, which happens more readily when the classifier only uses the original dataset features rather than combinations of them, and when the discrimination models are low-dimensional.

In this context, we aim to learn a set of classifiers in low-dimensional subspaces and a decision function which selects the subspace under which a test point is to be classified. Assume we are given a dataset $\{(x_1, y_1) \ldots (x_n, y_n)\} \in \mathcal{X}^n \times \{0,1\}^n$ and a class of discriminators $\mathcal{H}$. The model will contain a set $\Pi$ of subspaces of $\mathcal{X}$; $\Pi \subseteq \mathbf{\Pi}$, where $\mathbf{\Pi}$ is the set of all axis-aligned subspaces of the original feature space, the power set of the features. To each projection $\pi_i \in \Pi$ corresponds one discriminator from a given hypothesis space $h_i \in \mathcal{H}$. It will also contain a selection function $g : \mathcal{X} \to \Pi \times \mathcal{H}$, which yields, for a query point $x$, the projection/discriminator pair with which this point will be classified. The notation $\pi(x)$ refers to the projection of the point $x$ onto the subspace

$\pi$ while $h(\pi(x))$ represents the predicted label for $x$. Formally, we describe the model class as

$$
\begin{aligned}
\mathcal{M}_d \quad = \quad & \{\Pi = \{\pi : \pi \in \mathbf{\Pi}, dim(\pi) \leq d\}, \\
& H = \{h_i : h_i \in \mathcal{H}, h : \pi_i \to \mathcal{Y}, \forall i = 1 \dots |\Pi|\}, \\
& g \in \{f : \mathcal{X} \to \{1 \dots |\mathbf{\Pi}|\}\} \qquad .
\end{aligned}
$$

where $dim(\pi)$ presents the dimensionality of the subspace determined by the projection $\pi$. Note that only projections up to size $d$ will be considered, where $d$ is a parameter specific to the application. The set $H$ contains one discriminator from the hypothesis class $\mathcal{H}$ for each projection.

Intuitively, the aim is to minimize the expected classification error over $\mathcal{M}_d$, however, a notable modification is that the projection and, implicitly, the discriminator, are chosen according to the data point that needs to be classified. Given a query $x$ in the space $\mathcal{X}$, $g(x)$ will yield the subspace $\pi_{g(x)}$ onto which the query is projected and the discriminator $h_{g(x)}$ for it. Distinct test points can be handled using different combinations of subspaces and discriminators. We consider models that minimize 0/1 loss. Hence, the PRC problem can be stated as follows:

$$
M^* = \underset{M \in \mathcal{M}_d}{\arg\min} \, \mathbb{E}_{\mathcal{X},\mathcal{Y}} \Big[ y \neq h_{g(x)}(\pi_{g(x)}(x)) \Big]
$$

There are limitations to the type of selection function $g$ that can be learned. A simple example for which g can be recovered is a set of signal readings $x$ for which, if one of the readings $x_i$ exceeds a threshold $t_i$, the label can be predicted just based on $x_i$. A more complex one is a dataset containing regulatory variables, that is, for $x_i$ in the interval $[a_k, b_k]$ the label only depends on $(x_k^1 \dots x_k^{n_k})$ - datasets that fall into the latter category fulfill what we call the Subspace-Separability Assumption.

This paper proposes an algorithm called RECIP that solves the PRC problem for a class of nonparametric classifiers. We evaluate the method on artificial data to show that indeed it correctly identifies the underlying structure for data satisfying the Subspace-Separability Assumption. We show some case studies to illustrate how RECIP offers insight into applications requiring human intervention.

The use of dimensionality reduction techniques is a common preprocessing step in applications where the use of simplified classification models is preferable. Methods that learn linear combinations of features, such as Linear Discriminant Analysis, are not quite appropriate for the task considered here, since we prefer to natively rely on the dimensions available in the original feature space. Feature selection methods, such as e.g. lasso, are suitable for identifying sets of relevant features, but do not consider interactions between them. Our work better fits the areas of class dependent feature selection and context specific classification, highly connected to the concept of Transductive Learning [6]. Other context-sensitive methods are Lazy and Data-Dependent Decision Trees, [5] and [10] respectively. In Ting et al [14], the Feating submodel selection relies on simple attribute splits followed by fitting local predictors, though the algorithm itself is substantially different. Obozinski et al present a subspace selection method in the context of multitask learning [11]. Go et al propose a joint method for feature selection and subspace learning [7], however, their classification model is not particularly query specific. Alternatively, algorithms that transform complex or unintelligible models with user-friendly equivalents have been proposed [3, 2, 1, 8]. Algorithms specifically designed to yield understandable models are a precious few. Here we note a rule learning method described in [12], even though the resulting rules can make visualization difficult, while itemset mining [9] is not specifically designed for classification. Unlike those approaches, our method is designed to retrieve subsets of the feature space designed for use in a way that is complementary to the basic task at hand (classification) while providing query-specific information.

## 2   Recovering informative projections with RECIP

To solve PRC, we need means by which to ascertain which projections are useful in terms of discriminating data from the two classes. Since our model allows the use of distinct projections depending on the query point, it is expected that each projection would potentially benefit different areas of the feature space. $\mathcal{A}(\pi)$ refers to the area of the feature space where the projection $\pi$ is selected.

$$
\mathcal{A}(\pi) \quad = \quad \{x \in \mathcal{X} : \pi_{g(x)} = \pi\}
$$

The objective becomes

$$
\min_{M \in \mathcal{M}_d} E_{(\mathcal{X} \times \mathcal{Y})} \Big[ y \neq h_{g(x)}(\pi_{g(x)}(x)) \Big] \quad = \quad \min_{M \in \mathcal{M}_d} \sum_{\pi \in \Pi} p(\mathcal{A}(\pi)) \mathbb{E}\Big( y \neq h_{g(x)}(\pi_{g(x)}(x)) | x \in \mathcal{A}(\pi) \Big) \qquad .
$$

The expected classification error over $\mathcal{A}(\pi)$ is linked to the conditional entropy of $Y|X$. Fano's inequality provides a lower bound on the error while Feder and Merhav [4] derive a tight upper bound on the minimal error probability in terms of the entropy. This means that conditional entropy characterizes the potential of a subset of the feature space to separate data, which is more generic than simply quantifying classification accuracy for a specific discriminator.

In view of this connection between classification accuracy and entropy, we adapt the objective to:

$$\min_{M \in \mathcal{M}_d} \sum_{\pi \in \Pi} p(\mathcal{A}(\pi)) H(Y|\pi(X); X \in \mathcal{A}(\pi)) \tag{1}$$

The method we propose optimizes an empirical analog of (1) which we develop below and for which we will need the following result.

**Proposition 2.1.** *Given a continuous variable $X \in \mathcal{X}$ and a binary variable $Y$, where $X$ is sampled from the mixture model $f(x) = p(y = 0)f_0(x) + p(y = 1)f_1(x) = p_0 f_0(x) + p_1 f_1(x)$,*

$$\text{then} \quad H(Y|X) = -p_0 \log p_0 - p_1 \log p_1 - D_{KL}(f_0||f) - D_{KL}(f_1||f).$$

Next, we will use the nonparametric estimator presented in [13] for Tsallis $\alpha$-divergence. Given samples $U_i \sim \mathcal{U}$, with $i = 1, n$ and $V_j \sim \mathcal{V}$ with $j = 1, m$, the divergence is estimated as follows:

$$\hat{T}_\alpha(U||V) = \frac{1}{1-\alpha} \left[ \frac{1}{n} \sum_{i=1}^n \left( \frac{(n-1)\nu_k(U_i, U \setminus u_i)^d}{m\nu_k(U_i, V)^d} \right)^{1-\alpha} B(k, \alpha) - 1 \right], \tag{2}$$

where $d$ is the dimensionality of the variables $U$ and $V$ and $\nu_k(z, Z)$ represents the distance from $z$ to its $k^{th}$ nearest neighbor of the set of points $Z$. For $\alpha \approx 1$ and $n \to \infty$, $\hat{T}_\alpha(u||v) \approx D_{KL}(u||v)$.

## 2.1 Local estimators of entropy

We will now plug (2) in the formula obtained by Proposition 2.1 to estimate the quantity (1). We use the notation $X_0$ to represent the $n_0$ samples from X which have the labels Y equal to 0, and $X_1$ to represent the $n_1$ samples from X which have the labels set to 1. Also, $X_{y(x)}$ represents the set of samples that have labels equal to the label of x and $X_{\neg y(x)}$ the data that have labels opposite to the label of x.

$$\hat{H}(Y|X; X \in \mathcal{A}) = -H(p_0) - H(p_1) - \hat{T}(f_0^x||f^x) - \hat{T}(f_1^x||f^x) + C \qquad \alpha \approx 1$$

$$\hat{H}(Y|X; X \in \mathcal{A}) \propto \frac{1}{n_0} \sum_{i=1}^{n_0} I[x_i \in \mathcal{A}] \left( \frac{(n_0 - 1)\nu_k(x_i, X_0 \setminus x_i)^d}{n\nu_k(x_i, X \setminus x_i)^d} \right)^{1-\alpha}$$

$$+ \frac{1}{n_1} \sum_{i=1}^{n_1} I[x_i \in \mathcal{A}] \left( \frac{(n_1 - 1)\nu_k(x_i, X_1 \setminus x_i)^d}{n\nu_k(x_i, X \setminus x_i)^d} \right)^{1-\alpha}$$

$$\propto \frac{1}{n_0} \sum_{i=1}^{n_0} I[x_i \in \mathcal{A}] \left( \frac{(n_0 - 1)\nu_k(x_i, X_0 \setminus x_i)^d}{n\nu_k(x_i, X_1 \setminus x_i)^d} \right)^{1-\alpha}$$

$$+ \frac{1}{n_1} \sum_{i=1}^{n_1} I[x_i \in \mathcal{A}] \left( \frac{(n_1 - 1)\nu_k(x_i, X_1 \setminus x_i)^d}{n\nu_k(x_i, X_0 \setminus x_i)^d} \right)^{1-\alpha}$$

$$\propto \frac{1}{n} \sum_{i=1}^{n} I[x_i \in \mathcal{A}] \left( \frac{(n-1)\nu_k(x_i, X_{y(x_i)} \setminus x_i)^d}{n\nu_k(x_i, X_{\neg y(x_i)} \setminus x_i)^d} \right)^{1-\alpha}$$

The estimator for the entropy of the data that is classified with projection $\pi$ is as follows:

$$\hat{H}(Y|\pi(X); X \in \mathcal{A}(\pi)) \propto \frac{1}{n} \sum_{i=1}^{n} I[x_i \in \mathcal{A}(\pi)] \left( \frac{(n-1)\nu_k(\pi(x_i), \pi(X_{y(x_i)}) \setminus \pi(x_i))^d}{n\nu_k(\pi(x_i), \pi(X_{\neg y(x_i)}) \setminus x_i))^d} \right)^{1-\alpha} \tag{3}$$

From 3 and using the fact that $I[x_i \in \mathcal{A}(\pi)] = I[\pi_{g(x_i)} = \pi]$ for which we use the notation $I[g(x_i) \to \pi]$, we estimate the objective as

$$\min_{M \in \mathcal{M}_d} \sum_{\pi \in \Pi} \frac{1}{n} \sum_{i=1}^{n} I[g(x_i) \to \pi] \left( \frac{(n-1)\nu_k(\pi(x_i), \pi(X_{y(x_i)}) \setminus \pi(x_i))^d}{n\nu_k(\pi(x_i), \pi(X_{\neg y(x_i)}) \setminus x_i))^d} \right)^{1-\alpha} \tag{4}$$

Therefore, the contribution of each data point to the objective corresponds to a distance ratio on the projection $\pi^*$ where the class of the point is obtained with the highest confidence (data is separable in the neighborhood of the point). We start by computing the distance-based metric of each point on each projection of size up to d - there are $d^*$ such projections.

This procedure yields an extended set of features $Z$, which we name local entropy estimates:

$$Z_{ij} = \left( \frac{\nu_k(\pi_j(x_i), \pi_j(X_{y(x_i)}) \setminus \pi_j(x_i))}{\nu_k(\pi_j(x_i), \pi_j(X_{\neg y(x_i)}) \setminus \pi_j(x_i))} \right)^{d(1-\alpha)} \qquad \alpha \approx 1 \quad j \in \{1 \ldots d^*\} \qquad (5)$$

For each training data point, we compute the best distance ratio amid all the projections, which is simply $T_i = \min_{j \in [d^*]} Z_{ij}$.

The objective can be then further rewritten as a function of the entropy estimates:

$$\min_{M \in \mathcal{M}_d} \sum_{i=1}^{n} \sum_{\pi_j \in \Pi} I[g(x_i) \to \pi_j] Z_{ij} \qquad (6)$$

From the definition of T, it is also clear that

$$\min_{M \in \mathcal{M}_d} \sum_{i=1}^{n} \sum_{\pi_j \in \Pi} I[g(x_i) \to \pi_j] Z_{ij} \quad \geq \quad \sum_{i=1}^{n} T_i \,. \qquad (7)$$

## 2.2 Projection selection as a combinatorial problem

Considering form (6) of the objective, and given that the estimates $Z_{ij}$ are constants, depending only on the training set, the projection retrieval problem is reduced to finding $g$ for all training points, which will implicitly select the projection set of the model. Naturally, one might assume the best-performing classification model is the one containing all the axis-aligned subspaces. This model achieves the lower bound (7) for the training set. However, the larger the set of projections, the more values the function $g$ takes, and thus the problem of selecting the correct projection becomes more difficult. It becomes apparent that the number of projections should be somehow restricted to allow intepretability. Assuming a hard threshold of at most $t$ projections, the optimization (6) becomes an entry selection problem over matrix $Z$ where one value must be picked from each row under a limitation on the number of columns that can be used. This problem cannot be solved exactly in polynomial time. Instead, it can be formulated as an optimization problem under $\ell_1$ constraints.

## 2.3 Projection retrieval through regularized regression

To transform the projection retrieval to a regression problem we consider T, the minimum obtainable value of the entropy estimator for each point, as the output which the method needs to predict. Each row $i$ of the parameter matrix $B$ represents the degrees to which the entropy estimates on each projection contribute to the entropy estimator of point $x_i$. Thus, the sum over each row of $B$ is 1, and the regularization penalty applies to the number of non-zero columns in $B$.

$$\min_{B} ||T - (Z \odot B)J_{|\Pi|,1}||_2^2 + \lambda \sum_{i=1}^{d^*} [B_i \neq 0] \qquad (8)$$
$$\text{subject to} \qquad |B_k|_{\ell_1} = 1 \quad k = \overline{1,n}$$
$$\text{where} \qquad (Z \odot B)_{ij} = Z_{ij} + B_{ij} \text{ and } J \text{ is a matrix of ones.}$$

The problem with this optimization is that it is not convex. A typical walk-around of this issue is to use the convex relaxation for $B_i \neq 0$, that is $\ell_1$ norm. This would transform the penalized term to $\sum_{i=1}^{d^*} |B_i|_{\ell_1}$. However, $\sum_{i=1}^{d^*} |B_i|_{\ell_1} = \sum_{k=1}^{n} |B_k|_{\ell_1} = n$, so this penalty really has no effect. An alternative mechanism to encourage the non-zero elements in $B$ to populate a small number of columns is to add a penalty term in the form of $B\delta$, where $\delta$ is a $d^*$-size column vector with each element representing the penalty for a column in $B$. With no prior information about which subspaces are more informative, $\delta$ starts as an all-1 vector. An initial value for $B$ is obtained through the optimization (8). Since our goal is to handle data using a small number of projections, $\delta$ is then updated such that its value is lower for the denser columns in $B$. This update resembles the re-weighing in adaptive lasso. The matrix $B$ itself is updated, and this 2-step process continues until convergence of $\delta$. Once $\delta$ converges, the projections corresponding to the non-zero columns of $B$ are added to the model. The procedure is shown in Algorithm 1.

**Algorithm 1:** RECIP

$\delta = [1 \dots 1]$
**repeat**
$\quad b = \arg\min_B ||T - \sum_{i=1}^{|PI|} <Z, B>||_2^2 + \lambda|B\delta|_{\ell_1}$
$\quad\quad$ subject to $\quad |B_k|_{\ell_1} = 1 \quad\quad k = 1 \dots n$
$\quad \delta_k = |B_i|_{\ell_1} \quad\quad i = \dots d^*$ (update the differential penalty)
$\quad \delta = 1 - \frac{\delta}{|\delta|_{\ell_1}}$
**until** $\delta$ converges
**return** $\Pi = \left\{ \pi_i; \quad |B_i|_{\ell_1} > 0 \quad \forall i = 1 \dots d^* \right\}$

## 2.4 Lasso for projection selection

We will compare our algorithm to lasso regularization that ranks the projections in terms of their potential for data separability. We write this as an $\ell_1$-penalized optimization on the extended feature set Z, with the objective $T: \quad \min_\beta |T - Z\beta|_2 + \lambda|\beta|_{\ell_1}$. The lasso penalty to the coefficient vector encourages sparsity. For a high enough $\lambda$, the sparsity pattern in $\beta$ is indicative of the usefulness of the projections. The lasso on entropy contributions was not found to perform well as it is not query specific and will find one projection for all data. We improved it by allowing it to iteratively find projections - this robust version offers increased performance by reweighting the data thus focusing on different subsets of it. Although better than running lasso on entropy contributions, the robust lasso does not match RECIP's performance as the projections are selected gradually rather than jointly. Running the standard lasso on the original design matrix yields a set of relevant variables and it is not immediately clear how the solution would translate to the desired class.

## 2.5 The selection function

Once the projections are selected, the second stage of the algorithm deals with assigning the projection with which to classify a particular query point. An immediate way of selecting the correct projection starts by computing the local entropy estimator for each subspace with each class assignment. Then, we may select the label/subspace combination that minimizes the empirical entropy.

$$(i^*, \theta^*) = \arg\min_{i,\theta} \left( \frac{\nu_k(\pi_i(x), \pi_i(X_\theta))}{\nu_k(\pi_i(x), \pi_i(X_{\neg\theta}))} \right)^{dim(\pi_i)(1-\alpha)} \quad\quad i = 1 \dots d^* \quad, \quad \alpha \approx 1 \quad\quad (9)$$

# 3 Experimental results

In this section we illustrate the capability of RECIP to retrieve informative projections of data and their use in support of interpreting results of classification. First, we analyze how well RECIP can identify subspaces in synthetic data whose distribution obeys the subspace separability assumption (3.1). As a point of reference, we also present classification accuracy results (3.2) for both the synthetic data and a few real-world sets. This is to quantify the extent of the trade-off between fidelity of attainable classifiers and desired informativeness of the projections chosen by RECIP. We expect RECIP's classification performance to be slightly, but not substantially worse when compared to relevant classification algorithms trained to maximize classification accuracy. Finally, we present a few examples (3.3) of informative projections recovered from real-world data and their utility in explaining to human users the decision processes applied to query points.

A set of artificial data used in our experiments contains $q$ batches of data points, each of them made classifiable with high accuracy using one of available 2-dimensional subspaces $(x_k^1, x_k^2)$ with $k \in \{1 \dots q\}$. The data in batch $k$ also have the property that $x_k^1 > t_k$. This is done such that the group a point belongs to can be detected from $x_k^1$, thus $x_k^1$ is a regulatory variable. We control the amount of noise added to thusly created synthetic data by varying the proportion of noisy data points in each batch. The results below are for datasets with 7 features each, with number of batches $q$ ranging between 1 and 7. We kept the number of features specifically low in order to prevent excessive variation between any two sets generated this way, and to enable computing meaningful estimates of the expectation and variance of performance, while enabling creation of complicated data in which synthetic patterns may substantially overlap (using 7 features and 7 2-dimensional patterns implies that dimensions of at least 4 of the patterns will overlap). We implemented our method

to be scalable to the size and dimensionality of data and although for brevity we do not include a discussion of this topic here, we have successfully run RECIP against data with 100 features.

The parameter $\alpha$ is a value close to 1, because the Tsallis divergence converges to the KL divergence as alpha approaches 1. For the experiments on real-world data, $d$ was set to $n$ (all projections were considered). For the artificial data experiments, we reported results for $d = 2$ as they do not change significantly for $d >= 2$ because this data was synthesized to contain bidimensional informative projections. In general, if d is too low, the correct full set of projections will not be found, but it may be recovered partially. If d is chosen too high, there is a risk that a given selected projection p will contain irrelevant features compared to the true projection $p_0$. However, this situation only occurs if the noise introduced by these features in the estimators makes the entropy contributions on $p$ and $p_0$ statistically indistinguishable for a large subset of data. The users will choose d according to the desired/acceptable complexity of the resulting model. If the results are to be visually interpreted by a human, values of 2 or 3 are reasonable for $d$.

### 3.1 Recovering informative projections

Table 1 shows how well RECIP recovers the $q$ subspaces corresponding to the synthesized batches of data. We measure precision (proportion of the recovered projections that are known to be informative), and recall (proportion of known informative projections that are recovered by the algorithm). in Table 1, rows correspond to the number of distinct synthetic batches injected in data, $q$, and subsequent columns correspond to the increasing amounts of noise in data. We note that the observed precision is nearly perfect: the algorithm makes only 2 mistakes over the entire set of experiments, and those occur for highly noisy setups. The recall is nearly perfect as long as there is little overlap among the dimensions, that is when the injections do not interfere with each other. As the number of projections increases, the chances for overlap among the affected features also increase, which makes the data more confusing resulting on a gradual drop of recall until only about 3 or 4 of the 7 known to be informative subspaces can be recovered. We have also used lasso as described in 2.4 in an attempt to recover projections. This setup only manages to recover one of the informative subspaces, regardless of how the regularization parameter is tuned.

### 3.2 Classification accuracy

Table 2 shows the classification accuracy of RECIP, obtained using synthetic data. As expected, the observed performance is initially high when there are few known informative projections in data and it decreases as noise and ambiguity of the injected patterns increase.

Most types of ensemble learners would use a voting scheme to arrive at the final classification of a testing sample, rather than use a model selection scheme. For this reason, we have also compared predictive accuracy revealed by RECIP against a method based on majority voting among multiple candidate subspaces. Table 4 shows that the accuracy of this technique is lower than the accuracy of RECIP, regardless of whether the informative projections are recovered by the algorithm or assumed to be known a priori. This confirms the intuition that a selection-based approach can be more effective than voting for data which satisfies the subspace separability assumption.

For reference, we have also classified the synthetic data using K-Nearest-Neighbors algorithm using all available features at once. The results of that experiment are shown in Table 5. Since RECIP uses neighbor information, K-NN is conceptually the closest among the popular alternatives. Compared to RECIP, K-NN performs worse when there are fewer synthetic patterns injected in data to form informative projections. It is because some features used then by K-NN are noisy. As more features become informative, the K-NN accuracy improves. This example shows the benefit of a selective approach to feature space and using a subset of the most explanatory projections to support not only explanatory analyses but also classification tasks in such circumstances.

### 3.3 RECIP case studies using real-world data

Table 3 summarizes the RECIP and K-NN performance on UCI datasets. We also test the methods using Cell dataset containing a set of measurements such as the area and perimeter biological cells with separate labels marking cells subjected to treatment and control cells. In Vowel data, the nearest-neighbor approach works exceptionally well, even outperforming random forests (0.94 accuracy), which is an indication that all features are jointly relevant. For d lower than the number of features, RECIP picks projections of only one feature, but if there is no such limitation, RECIP picks the space of all the features as informative.

Table 1: Projection recovery for artificial datasets with $1 \ldots 7$ informative features and noise level $0 \ldots 0.2$ in terms of mean and variance of *Precision* and *Recall*. Mean/var obtained for each setting by repeating the experiment with datasets with different informative projections.

| | **PRECISION** | | | | | | | | | |
|---|---|---|---|---|---|---|---|---|---|---|
| | **Mean** | | | | | **Variance** | | | | |
| | *0* | *0.02* | *0.05* | *0.1* | *0.2* | *0* | *0.02* | *0.05* | *0.1* | *0.2* |
| *1* | 1 | 1 | 1 | 0.9286 | 0.9286 | 0 | 0 | 0 | 0.0306 | 0.0306 |
| *2* | 1 | 1 | 1 | 1 | 1 | 0 | 0 | 0 | 0 | 0 |
| *3* | 1 | 1 | 1 | 1 | 1 | 0 | 0 | 0 | 0 | 0 |
| *4* | 1 | 1 | 1 | 1 | 1 | 0 | 0 | 0 | 0 | 0 |
| *5* | 1 | 1 | 1 | 1 | 1 | 0 | 0 | 0 | 0 | 0 |
| *6* | 1 | 1 | 1 | 1 | 1 | 0 | 0 | 0 | 0 | 0 |
| *7* | 1 | 1 | 1 | 1 | 1 | 0 | 0 | 0 | 0 | 0 |
| | **RECALL** | | | | | | | | | |
| | **Mean** | | | | | **Variance** | | | | |
| | *0* | *0.02* | *0.05* | *0.1* | *0.2* | *0* | *0.02* | *0.05* | *0.1* | *0.2* |
| *1* | 1 | 1 | 1 | 1 | 1 | 0 | 0 | 0 | 0 | 0 |
| *2* | 1 | 1 | 1 | 1 | 1 | 0 | 0 | 0 | 0 | 0 |
| *3* | 1 | 1 | 0.9524 | 0.9524 | 1 | 0 | 0 | 0.0136 | 0.0136 | 0 |
| *4* | 0.9643 | 0.9643 | 0.9643 | 0.9643 | 0.9286 | 0.0077 | 0.0077 | 0.0077 | 0.0077 | 0.0128 |
| *5* | 0.7714 | 0.7429 | 0.8286 | 0.8571 | 0.7714 | 0.0163 | 0.0196 | 0.0049 | 0.0082 | 0.0278 |
| *6* | 0.6429 | 0.6905 | 0.6905 | 0.6905 | 0.6905 | 0.0113 | 0.0113 | 0.0272 | 0.0113 | 0.0113 |
| *7* | 0.6327 | 0.5918 | 0.5918 | 0.5714 | 0.551 | 0.0225 | 0.02 | 0.0258 | 0.0233 | 0.02 |

Table 2: RECIP Classification Accuracy on Artificial Data

| | **CLASSIFICATION ACCURACY** | | | | | | | | | |
|---|---|---|---|---|---|---|---|---|---|---|
| | **Mean** | | | | | **Variance** | | | | |
| | *0* | *0.02* | *0.05* | *0.1* | *0.2* | *0* | *0.02* | *0.05* | *0.1* | *0.2* |
| *1* | 0.9751 | 0.9731 | 0.9686 | 0.9543 | 0.9420 | 0.0000 | 0.0000 | 0.0000 | 0.0008 | 0.0007 |
| *2* | 0.9333 | 0.9297 | 0.9227 | 0.9067 | 0.8946 | 0.0001 | 0.0001 | 0.0001 | 0.0001 | 0.0001 |
| *3* | 0.9053 | 0.8967 | 0.8764 | 0.8640 | 0.8618 | 0.0004 | 0.0005 | 0.0016 | 0.0028 | 0.0007 |
| *4* | 0.8725 | 0.8685 | 0.8589 | 0.8454 | 0.8187 | 0.0020 | 0.0020 | 0.0019 | 0.0025 | 0.0032 |
| *5* | 0.8113 | 0.8009 | 0.8105 | 0.8105 | 0.7782 | 0.0042 | 0.0044 | 0.0033 | 0.0036 | 0.0044 |
| *6* | 0.7655 | 0.7739 | 0.7669 | 0.7632 | 0.7511 | 0.0025 | 0.0021 | 0.0026 | 0.0025 | 0.0027 |
| *7* | 0.7534 | 0.7399 | 0.7347 | 0.7278 | 0.7205 | 0.0034 | 0.0040 | 0.0042 | 0.0042 | 0.0045 |
| | **CLASSIFICATION ACCURACY - KNOWN PROJECTIONS** | | | | | | | | | |
| | **Mean** | | | | | **Variance** | | | | |
| | *0* | *0.02* | *0.05* | *0.1* | *0.2* | *0* | *0.02* | *0.05* | *0.1* | *0.2* |
| *1* | 0.9751 | 0.9731 | 0.9686 | 0.9637 | 0.9514 | 0.0000 | 0.0000 | 0.0000 | 0.0001 | 0.0000 |
| *2* | 0.9333 | 0.9297 | 0.9227 | 0.9067 | 0.8946 | 0.0001 | 0.0001 | 0.0001 | 0.0001 | 0.0001 |
| *3* | 0.9053 | 0.8967 | 0.8914 | 0.8777 | 0.8618 | 0.0004 | 0.0005 | 0.0005 | 0.0007 | 0.0007 |
| *4* | 0.8820 | 0.8781 | 0.8657 | 0.8541 | 0.8331 | 0.0011 | 0.0011 | 0.0014 | 0.0014 | 0.0020 |
| *5* | 0.8714 | 0.8641 | 0.8523 | 0.8429 | 0.8209 | 0.0015 | 0.0015 | 0.0018 | 0.0019 | 0.0023 |
| *6* | 0.8566 | 0.8497 | 0.8377 | 0.8285 | 0.8074 | 0.0014 | 0.0015 | 0.0016 | 0.0023 | 0.0021 |
| *7* | 0.8429 | 0.8371 | 0.8256 | 0.8122 | 0.7988 | 0.0015 | 0.0018 | 0.0018 | 0.0021 | 0.0020 |

Table 3: Accuracy of K-NN and RECIP

| *Dataset* | *KNN* | *RECIP* |
|---|---|---|
| Breast Cancer Wis | **0.8415** | 0.8275 |
| Breast Tissue | **1.0000** | **1.0000** |
| Cell | 0.7072 | **0.7640** |
| MiniBOONE* | **0.7896** | 0.7396 |
| Spam | **0.7680** | **0.7680** |
| Vowel | **0.9839** | **0.9839** |

In Spam data, the two most informative projections are 'Capital Length Total' (CLT)/'Capital Length Longest' (CLL) and CLT/'Frequency of word your' (FWY). Figure 1 shows these two projections, with the dots representing training points. The red dots represent points labeled as spam while the blue ones are non-spam. The circles are query points that have been assigned to be classified with the projection in which they are plotted. The green circles are correctly classified points, while the magenta circles - far fewer - are the incorrectly classified ones. Not only does the importance of text in capital letters make sense for a spam filtering dataset, but the points that select those projections are almost flawlessly classified. Additionally, assuming the user would need to attest the validity of classification for the first plot, he/she would have no trouble seeing that the circled data points are located in a region predominantly populated with examples of spam, so any non-spam entry appears suspicious. Both of the magenta-colored cases fall into this category, and they can be therefore flagged for further investigation.

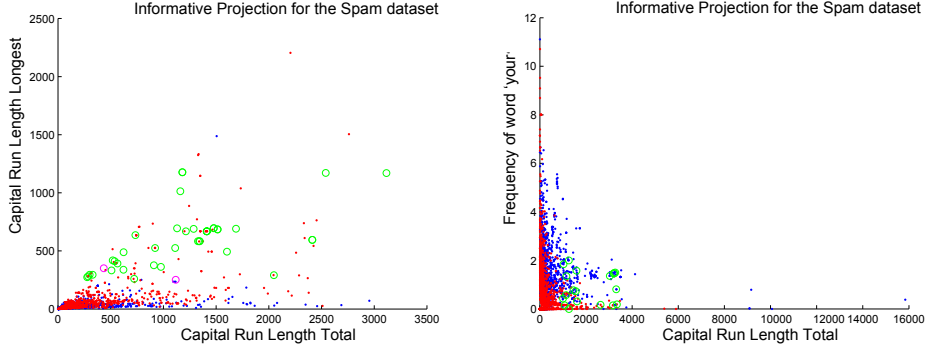

Figure 1: Spam Dataset Selected Subspaces

Table 4: Classification accuracy using RECIP-learned projections - or known projections, in the lower section - within a voting model instead of a selection model

| | CLASSIFICATION ACCURACY - VOTING ENSEMBLE | | | | | | | | | |
|---|---|---|---|---|---|---|---|---|---|---|
| | Mean | | | | | Variance | | | | |
| | 0 | 0.02 | 0.05 | 0.1 | 0.2 | 0 | 0.02 | 0.05 | 0.1 | 0.2 |
| 1 | 0.9751 | 0.9731 | 0.9686 | 0.9317 | 0.9226 | 0.0000 | 0.0000 | 0.0000 | 0.0070 | 0.0053 |
| 2 | 0.7360 | 0.7354 | 0.7331 | 0.7303 | 0.7257 | 0.0002 | 0.0002 | 0.0001 | 0.0002 | 0.0001 |
| 3 | 0.7290 | 0.7266 | 0.7163 | 0.7166 | 0.7212 | 0.0002 | 0.0002 | 0.0008 | 0.0006 | 0.0002 |
| 4 | 0.6934 | 0.6931 | 0.6932 | 0.6904 | 0.6867 | 0.0008 | 0.0008 | 0.0008 | 0.0008 | 0.0009 |
| 5 | 0.6715 | 0.6602 | 0.6745 | 0.6688 | 0.6581 | 0.0013 | 0.0014 | 0.0013 | 0.0014 | 0.0013 |
| 6 | 0.6410 | 0.6541 | 0.6460 | 0.6529 | 0.6512 | 0.0008 | 0.0007 | 0.0010 | 0.0006 | 0.0005 |
| 7 | 0.6392 | 0.6342 | 0.6268 | 0.6251 | 0.6294 | 0.0009 | 0.0011 | 0.0012 | 0.0012 | 0.0012 |
| | CLASSIFICATION ACCURACY - VOTING ENSEMBLE, KNOWN PROJECTIONS | | | | | | | | | |
| | Mean | | | | | Variance | | | | |
| | 0 | 0.02 | 0.05 | 0.1 | 0.2 | 0 | 0.02 | 0.05 | 0.1 | 0.2 |
| 1 | 0.9751 | 0.9731 | 0.9686 | 0.9637 | 0.9514 | 0.0000 | 0.0000 | 0.0000 | 0.0001 | 0.0000 |
| 2 | 0.7360 | 0.7354 | 0.7331 | 0.7303 | 0.7257 | 0.0002 | 0.0002 | 0.0001 | 0.0002 | 0.0001 |
| 3 | 0.7409 | 0.7385 | 0.7390 | 0.7353 | 0.7325 | 0.0010 | 0.0012 | 0.0010 | 0.0011 | 0.0010 |
| 4 | 0.7110 | 0.7109 | 0.7083 | 0.7067 | 0.7035 | 0.0041 | 0.0041 | 0.0042 | 0.0042 | 0.0043 |
| 5 | 0.7077 | 0.7070 | 0.7050 | 0.7034 | 0.7008 | 0.0015 | 0.0015 | 0.0015 | 0.0016 | 0.0016 |
| 6 | 0.6816 | 0.6807 | 0.6801 | 0.6790 | 0.6747 | 0.0008 | 0.0008 | 0.0008 | 0.0008 | 0.0009 |
| 7 | 0.6787 | 0.6783 | 0.6772 | 0.6767 | 0.6722 | 0.0008 | 0.0009 | 0.0009 | 0.0008 | 0.0008 |

Table 5: Classification accuracy for artificial data with the K-Nearest Neighbors method

| | CLASSIFICATION ACCURACY - KNN | | | | | | | | | |
|---|---|---|---|---|---|---|---|---|---|---|
| | Mean | | | | | Variance | | | | |
| | 0 | 0.02 | 0.05 | 0.1 | 0.2 | 0 | 0.02 | 0.05 | 0.1 | 0.2 |
| 1 | 0.7909 | 0.7843 | 0.7747 | 0.7652 | 0.7412 | 0.0002 | 0.0002 | 0.0002 | 0.0002 | 0.0002 |
| 2 | 0.7940 | 0.7911 | 0.7861 | 0.7790 | 0.7655 | 0.0001 | 0.0001 | 0.0001 | 0.0001 | 0.0001 |
| 3 | 0.7964 | 0.7939 | 0.7901 | 0.7854 | 0.7756 | 0.0000 | 0.0001 | 0.0001 | 0.0000 | 0.0000 |
| 4 | 0.7990 | 0.7972 | 0.7942 | 0.7904 | 0.7828 | 0.0001 | 0.0001 | 0.0001 | 0.0001 | 0.0001 |
| 5 | 0.8038 | 0.8024 | 0.8002 | 0.7970 | 0.7905 | 0.0001 | 0.0001 | 0.0001 | 0.0001 | 0.0001 |
| 6 | 0.8043 | 0.8032 | 0.8015 | 0.7987 | 0.7930 | 0.0001 | 0.0001 | 0.0001 | 0.0001 | 0.0001 |
| 7 | 0.8054 | 0.8044 | 0.8028 | 0.8004 | 0.7955 | 0.0001 | 0.0001 | 0.0001 | 0.0001 | 0.0001 |

# 4    Conclusion

This paper considers the problem of Projection Recovery for Classification. It is relevant in applications where the decision process must be easy to understand in order to enable human interpretation of the results. We have developed a principled, regression-based algorithm designed to recover small sets of low-dimensional subspaces that support interpretability. It optimizes the selection using individual data-point-specific entropy estimators. In this context, the proposed algorithm follows the idea of transductive learning, and the role of the resulting projections bears resemblance to high confidence regions known in conformal prediction models. Empirical results obtained using simulated and real-world data show the effectiveness of our method in finding informative projections that enable accurate classification while maintaining transparency of the underlying decision process.

**Acknowledgments**

This material is based upon work supported by the NSF, under Grant No. IIS-0911032.

# References

[1] Mark W. Craven and Jude W. Shavlik. Extracting Tree-Structured Representations of Trained Networks. In David S. Touretzky, Michael C. Mozer, and Michael E. Hasselmo, editors, *Advances in Neural Information Processing Systems*, volume 8, pages 24–30. The MIT Press, 1996.

[2] Pedro Domingos. Knowledge discovery via multiple models. *Intelligent Data Analysis*, 2:187–202, 1998.

[3] Eulanda M. Dos Santos, Robert Sabourin, and Patrick Maupin. A dynamic overproduce-and-choose strategy for the selection of classifier ensembles. *Pattern Recogn.*, 41:2993–3009, October 2008.

[4] M. Feder and N. Merhav. Relations between entropy and error probability. *Information Theory, IEEE Transactions on*, 40(1):259–266, January 1994.

[5] Jerome H. Friedman, Ron Kohavi, and Yeogirl Yun. Lazy decision trees, 1996.

[6] A. Gammerman, V. Vovk, and V. Vapnik. Learning by transduction. In *In Uncertainty in Artificial Intelligence*, pages 148–155. Morgan Kaufmann, 1998.

[7] Quanquan Gu, Zhenhui Li, and Jiawei Han. Joint feature selection and subspace learning, 2011.

[8] Bing Liu, Minqing Hu, and Wynne Hsu. Intuitive representation of decision trees using general rules and exceptions. In *Proceedings of Seventeeth National Conference on Artificial Intellgience (AAAI-2000), July 30 - Aug 3, 2000*, pages 615–620, 2000.

[9] Michael Mampaey, Nikolaj Tatti, and Jilles Vreeken. Tell me what i need to know: succinctly summarizing data with itemsets. In *Proceedings of the 17th ACM SIGKDD international conference on Knowledge discovery and data mining*, KDD '11, pages 573–581, New York, NY, USA, 2011. ACM.

[10] Mario Marchand and Marina Sokolova. Learning with decision lists of data-dependent features. *JOURNAL OF MACHINE LEARNING REASEARCH*, 6, 2005.

[11] Guillaume Obozinski, Ben Taskar, and Michael I. Jordan. Joint covariate selection and joint subspace selection for multiple classification problems. *Statistics and Computing*, 20(2):231–252, April 2010.

[12] Michael J. Pazzani, Subramani Mani, and W. Rodman Shankle. Beyond concise and colorful: Learning intelligible rules, 1997.

[13] B. Poczos and J. Schneider. On the estimation of alpha-divergences. *AISTATS*, 2011.

[14] Kai Ting, Jonathan Wells, Swee Tan, Shyh Teng, and Geoffrey Webb. Feature-subspace aggregating: ensembles for stable andunstable learners. *Machine Learning*, 82:375–397, 2011. 10.1007/s10994-010-5224-5.

